# Spike-Timing Dependent Plasticity and Mutual Information Maximization for a Spiking Neuron Model

**Taro Toyoizumi**[†‡],      **Jean-Pascal Pfister**[‡]

**Kazuyuki Aihara**[§] [*]      **Wulfram Gerstner**[‡]

† Department of Complexity Science and Engineering,
The University of Tokyo, 153-8505 Tokyo, Japan

‡ Ecole Polytechnique Fédérale de Lausanne (EPFL),
School of Computer and Communication Sciences and
Brain-Mind Institute, 1015 Lausanne, Switzerland

§ Graduate School of Information Science and Technology,
The University of Tokyo, 153-8505 Tokyo, Japan

taro@sat.t.u-tokyo.ac.jp,      jean-pascal.pfister@epfl.ch
aihara@sat.t.u-tokyo.ac.jp,      wulfram.gerstner@epfl.ch

## Abstract

We derive an optimal learning rule in the sense of mutual information maximization for a spiking neuron model. Under the assumption of small fluctuations of the input, we find a spike-timing dependent plasticity (STDP) function which depends on the time course of excitatory postsynaptic potentials (EPSPs) and the autocorrelation function of the postsynaptic neuron. We show that the STDP function has both positive and negative phases. The positive phase is related to the shape of the EPSP while the negative phase is controlled by neuronal refractoriness.

## 1   Introduction

Spike-timing dependent plasticity (STDP) has been intensively studied during the last decade both experimentally and theoretically (for reviews see [1, 2]). STDP is a variant of Hebbian learning that is sensitive not only to the spatial but also to the temporal correlations between pre- and postsynaptic neurons. While the exact time course of the STDP function varies between different types of neurons, the functional consequences of these differences are largely unknown. One line of modeling research takes a given STDP rule and analyzes the evolution of synaptic efficacies [3–5]. In this article, we take a different

---

[*]Alternative address: ERATO Aihara Complexity Modeling Project, JST, 45-18 Oyama, Shibuya-ku, 151-0065 Tokyo , Japan

approach and start from first principles. More precisely, we ask what is the optimal synaptic update rule so as to maximize the mutual information between pre- and postsynaptic neurons.

Previously information theoretical approaches to neural coding have been used to quantify the amount of information that a neuron or a neural network is able to encode or transmit [6–8]. In particular, algorithms based on the maximization of the mutual information between the output and the input of a network, also called infomax principle [9], have been used to detect the principal (or independent) components of the input signal, or to reduce the redundancy [10–12]. Although it is a matter of discussion whether neurons simply 'transmit' information as opposed to classification or task-specific processing [13], strategies based on information maximization provide a reasonable starting point to construct neuronal networks in an unsupervised, but principled manner.

Recently, using a rate neuron, Chechik applied information maximization to detect static input patterns from the output signal, and derived the optimal temporal learning window; the learning window has a positive part due to the effect of the postsynaptic potential and has flat negative parts with a length determined by the memory span [14].

In this paper, however, we employ a stochastic spiking neuron model to study not only the effect of postsynaptic potentials generated by synaptic input but also the effect of the refractory period of the postsynaptic neuron on the shape of the optimal learning window. We discuss the relation of mutual information and Fisher information for small input variance in Sec. 2. Optimization of the Fisher information by gradient ascent yields an optimal learning rule as shown in Sec. 3

## 2  Model assumptions

### 2.1  Neuron model

The model we are considering is a stochastic neuron with refractoriness. The instantaneous firing rate $\rho$ at time $t$ depends on the membrane potential $u(t)$ and refractoriness $R(t)$:

$$\rho(t) = g(\beta u(t))R(t), \tag{1}$$

where $g(\beta u) = g_0 \log_2[1 + e^{\beta u}]$ is a smoothed piecewise linear function with a scaling variable $\beta$ and a constant $g_0 = 85$Hz. The refractory variable is $R(t) = \frac{(t - \hat{t} - \tau_{\mathrm{abs}})^2}{\tau_{\mathrm{refr}}^2 + (t - \hat{t} - \tau_{\mathrm{abs}})^2}\Theta(t - \hat{t} - \tau_{\mathrm{abs}})$ and depends on the time elapsed since the last firing time $\hat{t}$, the absolute refractory period $\tau_{\mathrm{abs}} = 3$ ms, and the time constant of relative refractoriness $\tau_{\mathrm{refr}} = 10$ ms. The Heaviside step function $\Theta$ takes a value of 1 for positive arguments and zero otherwise. The postsynaptic potential depends on the input spike trains of $N$ presynaptic neurons. A presynaptic spike of neuron $i \in \{1, 2, \ldots, N\}$ emitted at time $t_i^f$ evokes a postsynaptic potential with time course $\epsilon(t - t_i^f)$. The total membrane potential is

$$u(t) = \sum_{i=1}^{N} w_i \sum_f \epsilon(t - t_i^f) = \sum_{i=1}^{N} w_i \int \epsilon(s) x_i(t - s) ds \tag{2}$$

where $x_i(t) = \sum_f \delta(t - t_i^f)$ denotes the spike train of the presynaptic neuron $i$. The above model is a special case of the spike response model with escape noise [2]. For vanishing refractoriness $\tau_{\mathrm{refr}} \to 0$ and $\tau_{\mathrm{abs}} \to 0$, the above model reduces to an inhomogeneous Poisson process.

For a given set of presynaptic spikes in an interval $[0, T]$, hence for a given time course of

membrane potential $\{u(t)|t \in [0,T]\}$, the model generates an output spike train

$$y(t) = \sum_f \delta(t - t^f) \qquad (3)$$

with firing times $\{t^f|f = 1, \ldots, n\}$ with a probability density

$$P(y|u) = \exp\left[\int_0^T (y(t) \log \rho(t) - \rho(t)) \, dt\right]. \qquad (4)$$

where $\rho(t)$ is given by Eq. (1), i.e., $\rho(t) = g(\beta u(t)) R(t)$. Since the refractory variable $R$ depends on the firing time $\hat{t}$ of the *previous* output spike, we sometimes write $\rho(t|\hat{t})$ instead of $\rho(t)$ in order to make this dependence explicit. Equation (4) can then be re-expressed in terms of the survivor function $S(t|\hat{t}) = e^{-\int_{\hat{t}}^t \rho(s|\hat{t})ds}$ and the interval distribution $Q(t|\hat{t}) = \rho(t|\hat{t})S(t|\hat{t})$ in a more transparent form:

$$P(y|u) = \left(\prod_{f=1}^n Q(t^f|t^{f-1})\right) S(T|t^n), \qquad (5)$$

where $t^0 = 0$ and $n$ is the number of postsynaptic spikes in $[0,T]$. In words, the probability that a specific output spike train $y$ occurs can be calculated from the interspike intervals $Q(t^f|t^{f-1})$ and the probability that the neuron 'survives' from the last spike at time $t^n$ to time $T$ without further firing.

## 2.2 Fisher information and mutual information

Let us consider input spike trains with stationary statistics. These input spike trains generate an input potential $u(t)$ with an average value $u_0$ and standard deviation $\sigma$. Assuming a weak dependence of $g$ on the membrane potential $u$, i.e., for small $\beta$, we expand $g$ around $g_0 = g(0)$ to obtain $g(\beta u(t)) = g_0 + g_0'\beta u(t) + g_0''[\beta u(t)]^2/2 + O(\beta^3)$ where $g_0$ is the value of $g$ in the absence of input and the next terms describe the influence of the input. Here and in the following, all calculations will be done to order $\beta^2$.

In the limit of small $\beta$, the mutual information is given by [15]

$$I(Y;X) = \frac{\beta^2}{2} \int_0^T dt \int_0^T dt' \Sigma(t - t') J_0(t - t') + O(\beta^3), \qquad (6)$$

with the autocovariance function of the membrane potential

$$\Sigma(t - t') = \langle \Delta u(t) \Delta u(t') \rangle_X, \qquad (7)$$

with $\Delta u(t) = u(t) - u_0$ and Fisher information

$$J_0(t - t') = -\left\langle \left.\frac{\partial^2 \log P(y|u)}{\partial \beta u(t) \partial \beta u(t')}\right|_{\beta=0} \right\rangle_{Y|\beta=0}, \qquad (8)$$

with $\langle \cdot \rangle_{Y|\beta=0} = \int \cdot P(y|\beta = 0)dy$ and $\langle \cdot \rangle_X = \int \cdot P(x)dx$. Note that the Fisher information (8) is to be evaluated at the constant $g_0$, i.e., at the value $\beta u = 0$, whereas the autocovariance in Eq. (7) is calculated with respect to the *mean* membrane potentital $u_0 = \langle u(t) \rangle_X$ which is in general different from zero. The derivation of (6) is based on the assumption that the variability of the output signal is small and $g(\beta u)$ does not deviate much from $g_0$, i.e., it corresponds to the regime of small signal-to-noise ratio. It is well known that the information capacity of the Gaussian channel is given by the log of the signal-to-noise ratio [16], and the mutual information is proportional to the

signal-to-noise ratio when it is small. The relation between the Fisher information, the mutual information, and optimal tuning curves has previously been established in the regime of large signal-to-noise ratio [17].

We introduce the following notation: Let $\mu_0 = \langle y(t) \rangle_{Y|\beta=0} = \langle \rho(t) \rangle_{Y|\beta=0}$ be the spontaneous firing rate in the absence of input and $\mu_0^{-1} \langle y(t)y(t') \rangle_{Y|\beta=0} = \delta(t-t') + \mu_0[1+\phi(t-t')]$ be the postsynaptic firing probability at time $t$ given a postsynaptic spike at $t'$, i.e., the autocorrelation function of $Y$. From the theory of stationary renewal processes [2]

$$\mu_0 = \left[ \int s\, Q_0(s)ds \right]^{-1},$$

$$\mu_0[1+\phi(s)] = Q_0(|s|) + \int Q_0(s')\mu_0[1+\phi(|s|-s')]\,\Theta(|s|-s')ds', \qquad (9)$$

where $Q_0(s) = g_0 R(s)e^{-g_0[(s-\tau_{\rm abs})-\tau_{\rm refr}\arctan(s-\tau_{\rm abs})/\tau_{\rm refr}]}$ is the interval distribution for constant $g = g_0$. The interval distribution vanishes during the absolute refractory time $\tau_{\rm abs}$; cf. Fig. 1.

(A)  (B)

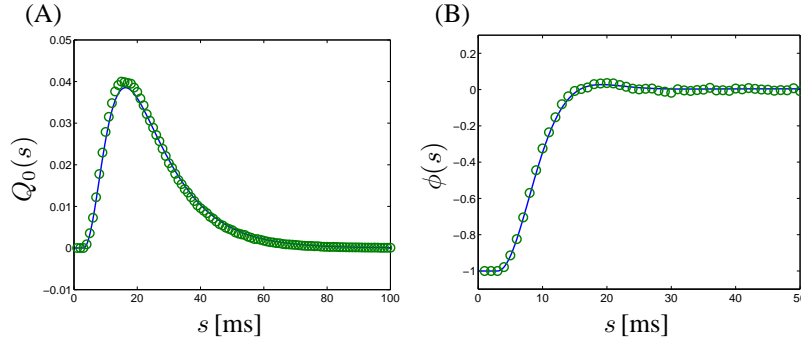

$\phi(s)$

$Q_0(s)$

Figure 1: Interspike interval distribution $Q_0$ and normalized autocorrelation function $\phi$. The circles show numerical results, the solid line the theory.

The Fisher information of (8) is calculated from (4) to be

$$J_0(t-t') = \delta(t-t') \left( \frac{g_0'}{g_0} \right)^2 \langle \rho_0(t) \rangle_{Y|\beta=0} \qquad (10)$$

with the instantaneous firing rate $\rho_0(t) = g_0 R(t)$. Hence the mutual information is

$$I(Y;X) = \frac{\beta^2}{2} \left( \frac{g_0'}{g_0} \right)^2 \int_0^T dt\, \mu_0 \sigma^2 \qquad (11)$$

$$= \frac{\beta^2}{2} \left( \frac{g_0'}{g_0} \right)^2 T\mu_0 \sigma^2. \qquad (12)$$

For an interpretation of Eq. (11) we note that $\sigma^2 = \Sigma(0)$ is the variance of the membrane potential and depends on the statistics of the *presynaptic* input whereas $\mu_0$ is the spontaneous firing rate which characterizes the output of the *postsynaptic* neuron. Hence, Equation (11) contains both pre- and postsynaptic factors.

## 3  Results: Optimal spike-timing dependent learning rule

In the previous section we have calculated the mutual information between presynaptic input spike trains and the output of the postsynaptic neuron under the assumption of small

fluctuations of $g$. The mutual information depends on parameters of the model neuron, in particular the synaptic weights that characterize the efficacy of the connections between pre- and postsynaptic neurons. In this section, we will optimize the mutual information by changing the synaptic weights in an appropriate fashion. To do so we will proceed in several steps.

First, based on gradient ascent we derive a batch learning rule of synaptic weights that maximizes the mutual information. In a second step, we transform the batch rule into an online rule that reduces to the batch version when averaged. Finally, in subsection 3.2, we will see that the online learning rule shares properties with STDP, in particular a biphasic dependence upon the relative timing of pre- and postsynaptic spikes.

## 3.1 Learning rule for spiking model neuron

In order to keep the analysis as simple as possible, we suppose that the input spike trains are independent Poisson trains, i.e., $\langle \Delta x_i(t) \Delta x_j(t') \rangle_X = \nu_i \delta(t - t') \delta_{ij}$, where $\Delta x_i(t) = x_i(t) - \nu_i$ with rate $\nu_i = \langle x_i(t) \rangle_X$. Then we obtain the variance of the membrane potential

$$\sigma^2 = \langle [\Delta u(t)]^2 \rangle_X = \epsilon_2 \sum_j w_j^2 \nu_j \tag{13}$$

with $\epsilon_2 = \int \epsilon^2(s) ds$.

Applying gradient ascent to (11) with an appropriate learning rate $\alpha$, we obtain the batch learning rule of synaptic weights as

$$\Delta w_i = \alpha \frac{\partial I(Y; X)}{\partial w_i} \approx \alpha \frac{\beta^2}{2} \left( \frac{g_0'}{g_0} \right)^2 \int_0^T dt\, \mu_0 \frac{\partial \sigma^2}{\partial w_i}. \tag{14}$$

The derivative of $\mu_0$ with respect to $w_i$ vanishes, since $\mu_0$ is the spontaneous firing rate in the absence of input. We note that both $\mu_0$ and $\sigma^2$ are defined by an ensemble averages, as is typical for a 'batch' rule.

While there are many candidates of online learning rule that give (14) on average, we are interested in rules that depend directly on neuronal spikes rather than mean rates. To proceed it is useful to write $\sigma^2 = \langle [\Delta u(t)]^2 \rangle_X$ with $\Delta u = \sum_i w_i \Delta \epsilon_i(t)$ where $\Delta \epsilon_i(t) = \int \epsilon(s) \Delta x_i(t - s) ds$. In this notation, one simple form of an online learning rule that depends on both the postsynaptic firing statistics and presynaptic autocorrelation is

$$\frac{dw_i}{dt} = \alpha \beta^2 \left( \frac{g_0'}{g_0} \right)^2 y(t) \Delta \epsilon_i(t) \Delta u(t), \tag{15}$$

Hence weights are updated with each postsynaptic spike with an amplitude proportional to an online estimate of the membrane potential variance calculated as the product of $\Delta u$ and $\Delta \epsilon_i$. Indeed, to order $\beta^0$, the input and the output spikes are independent; $\langle y(t) \Delta \epsilon_i(t) \Delta u(t) \rangle_{Y,X} = \langle y(t) \rangle_{Y|\beta=0} \langle \Delta \epsilon_i(t) \Delta u(t) \rangle_X$ and the average of (15) leads back to (14).

## 3.2 STDP function as a spike-pair effect

Application of the online learning rule (15) during a trial of duration $T$, yields a total change of the synaptic efficacy which depends on all the presynaptic spikes via the factor $\Delta \epsilon_i$; on the postsynaptic potential via the factor $\Delta u$; and on the postsynaptic spike train $y(t)$. In order to extract the spike pair effect evoked by a given presynaptic spike at $t_i^{pre}$ and a postsynaptic spike at $t^{post}$, we average over $x$ and $y$ given the pair of spikes. The spike pair effect up to the second order of $\beta$ is therefore described as

$$\Delta w_i(t^{post} - t_i^{pre}) = \alpha \beta^2 \left( \frac{g_0'}{g_0} \right)^2 \int_0^T dt \langle y(t) \rangle_{Y|t^{post}, \beta=0} \langle \Delta \epsilon_i(t) \Delta u(t) \rangle_{X|t_i^{pre}}, \tag{16}$$

where $\langle\cdot\rangle_{Y|t^{post},\beta=0} = \int dy \cdot P(y|t^{post},\beta=0)$ and $\langle\cdot\rangle_{X|t_i^{pre}} = \int dx \cdot P(x|t_i^{pre})$. Note that the leading factor of Eq. (16) is already of order $\beta^2$, so that all other factors have to be evaluated to order $\beta^0$. Suppressing all terms containing $\beta$, we obtain $P(y|t^{post},u) \approx P(y|t^{post},\beta=0)$ and from the Bayes formula $P(x|t_i^{pre},t^{post}) = \frac{P(t^{post}|x,t_i^{pre})}{\langle P(t^{post}|x,t_i^{pre})\rangle_{X|t_i^{pre}}}P(x|t_i^{pre}) \approx P(x|t_i^{pre})$.

In order to see the contribution of $t_i^{pre}$ and $t^{post}$, we think of separating the effects caused by spikes at $t_i^{pre}, t^{post}$ from the mean weight evolution caused by all other spikes. Therefore we insert $\langle y(t)\rangle_{Y|t^{post},\beta=0} = \delta(t-t^{post})+\mu_0[1+\phi(t-t^{post})]$ and $\langle\Delta\epsilon_i(t)\Delta u(t)\rangle_{X|t_i^{pre}} = w_i[\epsilon^2(t-t^{pre}) + \epsilon_2\nu_i]$ into Eq. (16) and decompose $\Delta w_i(t^{post} - t_i^{pre})$ into the following four terms: the drift term $\Delta w_i^0 = \alpha\beta^2\left(\frac{g_0'}{g_0}\right)^2 T\mu_0\epsilon_2 w_i\nu_i$ of the batch learning (14) that does not depend on $t_i^{pre}$ or $t^{post}$; the presynaptic component $\Delta w_i^{pre} = \alpha\beta^2\left(\frac{g_0'}{g_0}\right)^2 \mu_0\epsilon_2 w_i$ that is triggered by the presynaptic spike at $t_i^{pre}$; the postsynaptic component $\Delta w_i^{post} = \alpha\beta^2\left(\frac{g_0'}{g_0}\right)^2 \left[1 + \mu_0\int_0^T \phi(t-t^{post})dt\right]\epsilon_2 w_i\nu_i$ that is triggered by the postsynaptic spike at $t^{post}$; and the correlation component

$$\Delta w_i^{corr} = \alpha\beta^2\left(\frac{g_0'}{g_0}\right)^2 w_i\left[\epsilon^2(t^{post} - t_i^{pre}) + \mu_0\int_0^T \phi(t-t^{post})\epsilon^2(t-t_i^{pre})dt\right] \quad (17)$$

that depends on the difference of the pre- and postsynaptic spike timing.

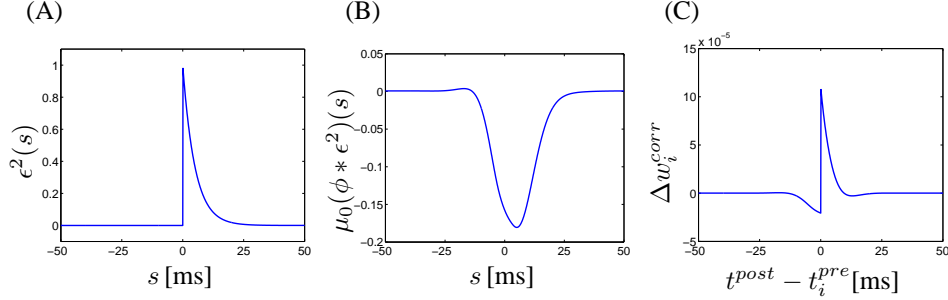

Figure 2: (A) The effect from EPSP: the first term in the square bracket of (17). (B) The effect from refractoriness: the second term in the square bracket of (17). (C) Temporal learning window $\Delta w_i^{corr}$ of (17).

In the following, we choose a simple exponential EPSP $\epsilon(t) = \Theta(s)e^{-s/\tau_u}$ with a time constant $\tau_u = 10$ ms. The parameters are $N = 100$, $\nu_i = 40$ Hz for all $i$, $w_i = (N\tau_u\nu_i)^{-1}$, $\alpha = 1$ and $\beta = 0.1$.

Figure 2 shows $\Delta w_i^{corr}$ of (17). The first term of (17) indicates the contribution of a presynaptic spike at $t_i^{pre}$ to increase the online estimation of membrane potential variance at time $t^{post}$, whereas the second term represents the effect of the refractory period on postsynaptic firing intensity, i.e., the normalized autocorrelation function convolved with the presynaptic contribution term. Due to the averaging of $\langle\cdot\rangle_{Y|t^{post},\beta=0}$ and $\langle\cdot\rangle_{X|t_i^{pre}}$ in (16), this optimal temporal learning window is local in time; we do not need to impose a memory span [14] to restrict the negative part of the learning window.

Figure 3 compares $\Delta w_i$ of (16) with numerical simulations of (15). We note a good agreement between theory and simulation. We recall, that all calculations, and hence the STDP function of (17) are valid for small $\beta$, i.e., for small fluctuation of $g$.

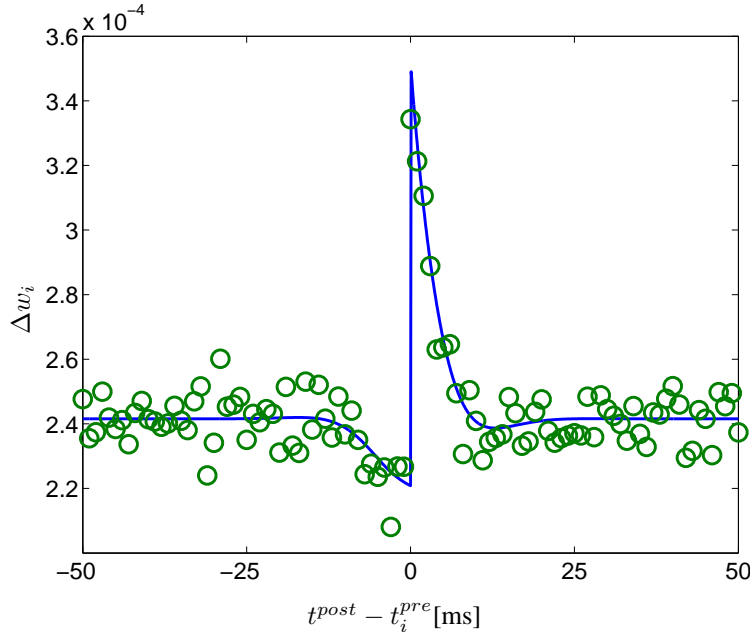

$t^{post} - t_i^{pre}$[ms]

$$t^{post} - t_i^{pre}\text{[ms]}$$

Figure 3: The comparison of the analytical result of (16) ( solid line ) and the numerical simulation of the online learning rule (15) ( circles ). For the simulation, the conditional average $\langle \Delta w_i \rangle_{X,Y|t_i^{pre},t^{post}}$ is evaluated by integrating $\frac{dw_i}{dt}$ over 200 ms around spike pairs with the given interval $t^{post} - t_i^{pre}$;

## 4  Conclusion

It is important for neurons especially in primary sensory systems to send information from previous processing circuits to neurons in other areas while capturing the essential features of its input. Mutual information is a natural quantity to be maximized from this perspective. We introduced an online learning rule for synaptic weights that increases information transmission for small input fluctuation. Introduction of the temporal properties of the target neuron enables us to analyze the temporal properties of the learning rule required to increase the mutual information. Consequently, the temporal learning window is given in terms of the time course of EPSPs and the autocorrelation function of the postsynaptic neuron. In particular, neuronal refractoriness plays a major role and yields the negative part of the learning window. Though we restrict our analysis here to excitatory synapses with independent spike trains, it is straightforward to generalize the approach to a mixture of excitatory and inhibitory neurons with weakly correlated spike trains as long as the synaptic weights are small enough. The analytically derived temporal learning window is similar to the experimentally observed bimodal STDP window [1]. Since the effective time course of EPSPs and the autocorrelation function of output spike trains vary from one part of the brain to another, it is important to compare those functions with the temporal learning window in biological settings.

**Acknowledgments**

T.T. is supported by the Japan Society for the Promotion of Science and a Grant-in-Aid for JSPS Fellows; J.-P.P. is supported by the Swiss National Science Foundation. We thank Y. Aviel for discussions.

# References

[1] G. Bi and M. Poo. Synaptic modification of correlated activity: Hebb's postulate revisited. *Annu. Rev. Neurosci.*, 24:139–166, 2001.

[2] W. Gerstner and W. M. Kistler. *Spiking Neuron Models*. Cambridge University Press, 2002.

[3] R. Kempter, W. Gerstner, and J. L. van Hemmen. Hebbian learning and spiking neurons. *Phys. Rev. E*, 59:4498–4514, 1999.

[4] W. Gerstner and W. M. Kistler. Mathematical formulations of hebbian learning. *Biol. Cybern.*, 87:404–415, 2002.

[5] R. Gütig, R. Aharonov, S. Rotter, and H. Sompolinsky. Learning input correlations through nonlinear temporally asymmetric hebbian plasticity. *J. Neurosci.*, 23(9):3697–3714, 2003.

[6] R. B. Stein. The information capacity of nerve cells using a frequency code. *Biophys. J.*, 7:797–826, 1967.

[7] W. Bialek, F. Rieke, R. de Ruyter van Stevenick, and D. Warland. Reading a neural code. *Science*, 252:1854–1857, 1991.

[8] F. Rieke, D. Warland, R. R. van Steveninck, and W. Bialek. *Spikes*. MIT Press, 1997.

[9] R. Linsker. Self-organization in a perceptual network. *Computer*, 21:105–117, 1988.

[10] J-P. Nadal and N. Parga. Nonlinear neurons in the low-noise limit: a factorial code maximizes information transfer. *Network: Comput.Neural Syst.*, 5:565–581, 1994.

[11] J-P Nadal, N. Brunel, and N Parga. Nonlinear feedforward networks with stochastic outputs: infomax implies redundancy reduction. *Network: Comput. Neural Syst.*, 9:207–217, 1998.

[12] A. J. Bell and T. Sejnowski. An information-maximization approach to blind separation and blind deconvolution. *Neural Comput.*, 7(6):1004–1034, 1995.

[13] J. J. Hopfield. Encoding for computation: recognizing brief dynamical patterns by exploiting effects of weak rhythms on action-potential timing. *Proc. Natl. Acad. Sci. USA*, 101(16):6255–6260, 2004.

[14] G. Checkik. Spike-timing-dependent plasticity and relevant mutual information maximization. *Neural Comput.*, 15:1481–1510, 2003.

[15] V. V. Prelov and E. C. van der Meulen. An asymptotic expression for the information and capacity of a multidimensional channel with weak input signals. *IEEE. Trans. Inform. Theory*, 39(5):1728–1735, 1993.

[16] T. M. Cover and J. A. Thomas. *Elements of Information Theory*. New York: Wiley, 1991.

[17] N. Brunel and J-P. Nadal. Mutual information, fisher information, and population coding. *Neural Comput.*, 10:1731–1757, 1998.
